# A Dynamic HMM for On–line Segmentation of Sequential Data

**Jens Kohlmorgen**[*]
Fraunhofer FIRST.IDA
Kekuléstr. 7
12489 Berlin, Germany
*jek@first.fraunhofer.de*

**Steven Lemm**
Fraunhofer FIRST.IDA
Kekuléstr. 7
12489 Berlin, Germany
*lemm@first.fraunhofer.de*

## Abstract

We propose a novel method for the analysis of sequential data that exhibits an inherent mode switching. In particular, the data might be a non-stationary time series from a dynamical system that switches between multiple operating modes. Unlike other approaches, our method processes the data incrementally and without any training of internal parameters. We use an HMM with a dynamically changing number of states and an on-line variant of the Viterbi algorithm that performs an unsupervised segmentation and classification of the data on-the-fly, i.e. the method is able to process incoming data in real-time. The main idea of the approach is to track and segment changes of the probability density of the data in a sliding window on the incoming data stream. The usefulness of the algorithm is demonstrated by an application to a switching dynamical system.

## 1  Introduction

Abrupt changes can occur in many different real-world systems like, for example, in speech, in climatological or industrial processes, in financial markets, and also in physiological signals (EEG/MEG). Methods for the analysis of time-varying dynamical systems are therefore an important issue in many application areas. In [12], we introduced the annealed competition of experts method for time series from non-linear switching dynamics, related approaches were presented, e.g., in [2, 6, 9, 14]. For a brief review of some of these models see [5], a good introduction is given in [3].

We here present a different approach in two respects. First, the segmentation does not depend on the predictability of the system. Instead, we merely estimate the density distribution of the data and track its changes. This is particularly an improvement for systems where data is hard to predict, like, for example, EEG recordings [7] or financial data. Second, it is an on-line method. An incoming data stream is processed incrementally while keeping the computational effort limited by a fixed

---

[*]http://www.first.fraunhofer.de/~jek

upper bound, i.e. the algorithm is able to perpetually segment and classify data streams with a fixed amount of memory and CPU resources. It is even possible to continuously monitor measured data in *real-time*, as long as the sampling rate is not too high.[1] The main reason for achieving a high on-line processing speed is the fact that the method, in contrast to the approaches above, does not involve any training, i.e. iterative adaptation of parameters. Instead, it optimizes the segmentation on-the-fly by means of dynamic programming [1], which thereby results in an automatic correction or fine-tuning of previously estimated segmentation bounds.

## 2 The segmentation algorithm

We consider the problem of continuously segmenting a data stream on-line and simultaneously labeling the segments. The data stream is supposed to have a *sequential* or *temporal structure* as follows: it is supposed to consist of consecutive blocks of data in such a way that the data points in each block originate from the same underlying distribution. The segmentation task is to be performed in an unsupervised fashion, i.e. without any a-priori given labels or segmentation bounds.

### 2.1 Using pdfs as features for segmentation

Consider $\vec{y}_1, \vec{y}_2, \vec{y}_3, \ldots$, with $\vec{y}_t \in R^n$, an incoming data stream to be analyzed. The sequence might have already passed a pre-processing step like filtering or sub-sampling, as long as this can be done on-the-fly in case of an on-line scenario. As a first step of further processing, it might then be useful to exploit an idea from dynamical systems theory and *embed* the data into a higher-dimensional space, which aims to reconstruct the state space of the underlying system,

$$\vec{x}_t = (\vec{y}_t, \vec{y}_{t-\tau}, \ldots, \vec{y}_{t-(m-1)\tau}). \tag{1}$$

The parameter $m$ is called the embedding dimension and $\tau$ is called the delay parameter of the embedding. The dimension of the vectors $\vec{x}_t$ thus is $d = m\,n$. The idea behind embedding is that the measured data might be a potentially non-linear projection of the systems state or phase space. In any case, an embedding in a higher-dimensional space might help to resolve structure in the data, a property which is exploited, e.g., in scatter plots. After the embedding step one might perform a sub-sampling of the embedded data in order to reduce the amount of data for real-time processing.[2] Next, we want to track the density distribution of the embedded data and therefore estimate the probability density function (pdf) in a sliding window of length $W$. We use a standard density estimator with multivariate Gaussian kernels [4] for this purpose, centered on the data points[3] in the window $\{\vec{x}_{t-w}\}_{w=0}^{W-1}$,

$$p_t(\mathbf{x}) = \frac{1}{W} \sum_{w=0}^{W-1} \frac{1}{(2\pi\sigma^2)^{d/2}} \exp\left(-\frac{(\mathbf{x} - \vec{x}_{t-w})^2}{2\sigma^2}\right). \tag{2}$$

The kernel width $\sigma$ is a smoothing parameter and its value is important to obtain a good representation of the underlying distribution. We propose to choose $\sigma$ proportional to the mean distance of each $\vec{x}_t$ to its first $d$ nearest neighbors, averaged over a sample set $\{\vec{x}_t\}$.

## 2.2 Similarity of two pdfs

Once we have sampled enough data points to compute the first pdf according to eq. (2), we can compute a new pdf with each new incoming data point. In order to quantify the difference between two such functions, $f$ and $g$, we use the squared $L_2$-Norm, also called *integrated squared error* (ISE), $d(f, g) = \int (f - g)^2 \, d\mathbf{x}$, which can be calculated analytically if $f$ and $g$ are mixtures of Gaussians as in our case of pdfs estimated from data windows,

$$d\left(p_{\bar{t}}(\mathbf{x}), p_t(\mathbf{x})\right) = \frac{1}{W^2 \left(4\pi\sigma^2\right)^{d/2}} \sum_{w,v=0}^{W-1} \left[ \exp\left(-\frac{(\vec{x}_{\bar{t}-w} - \vec{x}_{\bar{t}-v})^2}{4\sigma^2}\right) \right.$$
$$\left. -2 \exp\left(-\frac{(\vec{x}_{\bar{t}-w} - \vec{x}_{t-v})^2}{4\sigma^2}\right) + \exp\left(-\frac{(\vec{x}_{t-w} - \vec{x}_{t-v})^2}{4\sigma^2}\right)\right] \quad (3)$$

## 2.3 The HMM in the off-line case

Before we can discuss the on-line variant, it is necessary to introduce the HMM and the respective off-line algorithm first. For a given a data sequence, $\{\vec{x}_t\}_{t=1}^T$, we can obtain the corresponding sequence of pdfs $\{p_t(\mathbf{x})\}_{t\in\mathcal{S}}$, $\mathcal{S} = \{W, ..., T\}$, according to eq. (2). We now construct a hidden Markov model (HMM) where each of these pdfs is represented by a state $s \in \mathcal{S}$, with $\mathcal{S}$ being the *set of states* in the HMM. For each state $s$, we define a continuous *observation probability distribution*,

$$p(p_t(\mathbf{x}) \mid s) = \frac{1}{\sqrt{2\pi}\,\varsigma} \exp\left(-\frac{d(p_s(\mathbf{x}), p_t(\mathbf{x}))}{2\varsigma^2}\right), \quad (4)$$

for observing a pdf $p_t(\mathbf{x})$ in state $s$. Next, the *initial state distribution* $\{\pi_s\}_{s\in\mathcal{S}}$ of the HMM is given by the uniform distribution, $\pi_s = 1/N$, with $N = |\mathcal{S}|$ being the number of states. Thus, each state is a-priori equally probable. The HMM *transition matrix*, $A = (p_{ij})_{i,j\in\mathcal{S}}$, determines each probability to switch from a state $s_i$ to a state $s_j$. Our aim is to find a representation of the given sequence of pdfs in terms of a sequence of a small number of representative pdfs, that we call prototypes, which moreover exhibits only a small number of prototype changes. We therefore define $A$ in such a way that transitions to the same state are $k$ times more likely than transitions to any of the other states,

$$p_{ij} = \begin{cases} \frac{k}{k+N-1} & ; \text{if } i = j \\ \frac{1}{k+N-1} & ; \text{if } i \neq j \end{cases} \quad (5)$$

This completes the definition of our HMM. Note that this HMM has only two free parameters, $k$ and $\varsigma$. The well-known Viterbi algorithm [13] can now be applied to the above HMM in order to compute the optimal – i.e. the most likely – state sequence of prototype pdfs that might have generated the given sequence of pdfs. This state sequence represents the segmentation we are aiming at. We can compute the most likely state sequence more efficiently if we compute it in terms of costs, $c = -\log(p)$, instead of probabilities $p$, i.e. instead of computing the maximum of the likelihood function $L$, we compute the minimum of the cost function, $-\log(L)$, which yields the optimal state sequence as well. In this way we can replace products by sums and avoid numerical problems [13]. In addition to that, we can further simplify the computation for the special case of our particular HMM architecture, which finally results in the following algorithm:

For each time step, $t = W, ..., T$, we compute for all $s \in \mathcal{S}$ the cost $c_s(t)$ of the optimal state sequence from $W$ to $t$, subject to the constraint that it ends in state $s$ at

time $t$. We call these constrained optimal sequences $c$-paths and the unconstrained optimum $o^*$-path. The iteration can be formulated as follows, with $d_{s,t}$ being a short hand for $d(p_s(\mathbf{x}), p_t(\mathbf{x}))$ and $\delta_{s,\bar{s}}$ denoting the Kronecker delta function:

Initialization, $\forall s \in \mathcal{S}$:

$$c_s(W) := d_{s,W},  \tag{6}$$

Induction, $\forall s \in \mathcal{S}$:

$$c_s(t) := d_{s,t} + \min_{\bar{s} \in \mathcal{S}} \left\{ c_{\bar{s}}(t-1) + C\left(1 - \delta_{s,\bar{s}}\right) \right\}, \quad \text{for } t = W+1, ..., T,  \tag{7}$$

Termination:

$$o^* := \min_{s \in \mathcal{S}} \left\{ c_s(T) \right\}.  \tag{8}$$

The regularization constant $C$, which is given by $C = 2\varsigma^2 \log(k)$ and thus subsumes our two free HMM parameters, can be interpreted as transition cost for switching to a new state in the path.[4] The optimal prototype sequence with minimal costs $o^*$ for the complete series of pdfs, which is determined in the last step, is obtained by logging and updating the $c$-paths for all states $s$ during the iteration and finally choosing the one with minimal costs according to eq. (8).

## 2.4   The on-line algorithm

In order to turn the above segmentation algorithm into an on-line algorithm, we must restrict the incremental update in eq. (7), such that it only uses pdfs (and therewith states) from past data points. We neglect at this stage that memory and CPU resources are limited.

Suppose that we have already processed data up to $T - 1$. When a new data point $\vec{y}_T$ arrives at time $T$, we can generate a new embedded vector $\vec{x}_T$ (once we have sampled enough initial data points for the embedding), we have a new pdf $p_T(\mathbf{x})$ (once we have sampled enough embedded vectors $\vec{x}_t$ for the first pdf window), and thus we have given a new HMM state. We can also readily compute the distances between the new pdf and all the previous pdfs, $d_{T,t}$, $t < T$, according to eq. (3). A similarly simple and straightforward update of the costs, the $c$-paths and the optimal state sequence is only possible, however, if we neglect to consider potential $c$-paths that would have contained the new pdf as a prototype in previous segments. In that case we can simply reuse the $c$-paths from $T - 1$. The on-line update at time $T$ for these restricted paths, that we henceforth denote with a tilde, can be performed as follows:

For $T = W$, we have $\tilde{c}_W(W) := \tilde{o}^*(W) := d_{W,W} = 0$. For $T > W$:

1. Compute the cost $\tilde{c}_T(T-1)$ for the new state $s = T$ at time $T-1$:
   For $t = W, ..., T-1$, compute

$$\tilde{c}_T(t) := d_{T,t} + \begin{cases} 0 & , \text{ if } \quad t = W \\ \min\left\{\tilde{c}_T(t-1); \ \tilde{o}^*(t-1) + C\right\} , & \text{ else} \end{cases}  \tag{9}$$

   and update

$$\tilde{o}^*(t) := \tilde{c}_T(t), \quad \text{if} \quad \tilde{c}_T(t) < \tilde{o}^*(t).  \tag{10}$$

   Here we use all previous optimal segmentations $\tilde{o}^*(t)$, so we don't need to keep the complete matrix $(\tilde{c}_s(t))_{s,t \in \mathcal{S}}$ and repeatedly compute the minimum

over all states. However, we must store and update the history of optimal segmentations $\tilde{o}^*(t)$.

2. Update from $T-1$ to $T$ and compute $\tilde{c}_s(T)$ for all states $s \in \mathcal{S}$ obtained so far, and also get $\tilde{o}^*(T)$: For $s = W, ..., T$, compute

$$\tilde{c}_s(T) := d_{s,T} + \min\left\{\tilde{c}_s(T-1); \tilde{o}^*(T-1) + C\right\} \tag{11}$$

and finally get the cost of the optimal path

$$\tilde{o}^*(T) := \min_{s \in \mathcal{S}}\left\{\tilde{c}_s(T)\right\}. \tag{12}$$

As for the off-line case, the above algorithm only shows the update equations for the *costs* of the $\tilde{c}$- and $\tilde{o}^*$-paths. The associated state sequences must be logged simultaneously during the computation. Note that this can be done by just storing the sequence of switching points for each path. Moreover, we do not need to keep the full matrix $(\tilde{c}_s(t))_{s,t \in \mathcal{S}}$ for the update, the most recent column is sufficient.

So far we have presented the incremental version of the segmentation algorithm. This algorithm still needs an amount of memory and CPU time that is increasing with each new data point. In order to limit both resources to a fixed amount, we must remove old pdfs, i.e. old HMM states, at some point. We propose to do this by discarding all states with time indices smaller or equal to $s$ each time the path associated with $\tilde{c}_s(T)$ in eq. (11) exhibits a switch *back* from a more recent state/pdf to the currently considered state $s$ as a result of the min-operation in eq. (11). In the above algorithm this can simply be done by setting $W := s + 1$ in that case, which also allows us to discard the corresponding old $\tilde{c}_{\bar{s}}(T)$- and $\tilde{o}^*(t)$-paths, for all $\bar{s} \leq s$ and $t < s$. In addition, the "if $t = W$" initialization clause in eq. (9) must be ignored after the first such cut and the $\tilde{o}^*(W-1)$-path must therefore still be kept to compute the else-part also for $t = W$ now. Moreover, we do not have $\tilde{c}_T(W-1)$ and we therefore assume $\min\{\tilde{c}_T(W-1); \tilde{o}^*(W-1) + C\} = \tilde{o}^*(W-1) + C$ (in eq. (9)).

The explanation for this is as follows: A switch back in eq. (11) indicates that a new data distribution is established, such that the $\tilde{c}$-path that ends in a pdf state $s$ from an old distribution routes its path through one of the more recent states that represent the new distribution, which means that this has lower costs despite of the incurred additional transition. Vice versa, a newly obtained pdf is unlikely to properly represent the previous mode then, which justifies our above assumption about $\tilde{c}_T(W-1)$. The effect of the proposed cut-off strategy is that we discard paths that end in pdfs from old modes but still allow to find the optimal pdf prototype within the current segment.

Cut-off conditions occur shortly after mode changes in the data and cause the removal of HMM states with pdfs from old modes. However, if no mode change takes place in the incoming data sequence, no states will be discarded. We therefore still need to set a fixed upper limit $\kappa$ for the number of candidate paths/pdfs that are simultaneously under consideration if we only have limited resources available. When this limit is reached because no switches are detected, we must successively discard the oldest path/pdf stored, which finally might result in choosing a sub-optimal prototype for that segment however. Ultimately, a continuous discarding even *enforces* a change of prototypes after $2\kappa$ time steps if no switching is induced by the data until then. The buffer size $\kappa$ should therefore be as large as possible. In any case, the buffer overflow condition can be recorded along with the segmentation, which allows us to identify such artificial switchings.

## 2.5 The labeling algorithm

A labeling algorithm is required to identify segments that represent the same underlying distribution and thus have similar pdf prototypes. The labeling algorithm generates labels for the segments and assigns identical labels to segments that are similar in this respect. To this end, we propose a relatively simple on-line clustering scheme for the prototypes, since we expect the prototypes obtained from the same underlying distribution to be already well-separated from the other prototypes as a result of the segmentation algorithm. We assign a new label to a segment if the distance of its associated prototype to all preceding prototypes exceeds a certain threshold $\theta$, and we assign the existing label of the closest preceding prototype otherwise. This can be written as

$$l(R) = \begin{cases} newlabel, & \textbf{if } \min_{1 \leq r < R} \left\{ d(p_{t(r)}(\mathbf{x}), p_{t(R)}(\mathbf{x})) \right\} > \theta \\ l\left( \text{indexmin}_{1 \leq r < R} \left\{ d(p_{t(r)}(\mathbf{x}), p_{t(R)}(\mathbf{x})) \right\} \right), & \textbf{else;} \end{cases} \tag{13}$$

with the initialization $l(1) = newlabel$. Here, $r = 1, ..., R$, denotes the enumeration of the segments obtained so far, and $t(\cdot)$ denotes the mapping to the index of the corresponding pdf prototype. Note that the segmentation algorithm might replace *a number of* recent pdf prototypes (and also recent segmentation bounds) during the on-line processing in order to optimize the segmentation each time new data is presented. Therefore, a relabeling of all segments that have changed is necessary in each update step of the labeler.

As for the hyperparameters $\sigma$ and $C$, we developed an algorithm that computes a suitable value for $\theta$ from a sample set $\{\vec{x}_t\}$. We refer to our forthcoming publication [8].

## 3 Application

We illustrate our approach by an application to a time series from switching dynamics based on the Mackey-Glass delay differential equation,

$$\frac{dx(t)}{dt} = -0.1x(t) + \frac{0.2x(t - t_d)}{1 + x(t - t_d)^{10}}. \tag{14}$$

Eq. (14) describes a high-dimensional chaotic system that was originally introduced as a model of blood cell regulation [10]. In our example, four stationary operating modes, A, B, C, and D, are established by using different delays, $t_d = 17, 23, 30,$ and 35, respectively. The dynamics operates stationary in one mode for a certain number of time steps, which is chosen at random between 200 and 300 (referring to sub-sampled data with a step size $\Delta = 6$). It then randomly switches to one of the other modes with uniform probability. This procedure is repeated 15 times, it thus generates a switching chaotic time series with 15 stationary segments. We then added a relatively large amount of "measurement" noise to the series: zero-mean Gaussian noise with a standard deviation of 30% of the standard deviation of the original series.

The on-line segmentation algorithm was then applied to the noisy data, i.e. processing was performed on-line although the full data set was already available in this case. The scalar time series was embedded on-the-fly by using $m = 6$ and $\tau = 1$ (on the sub-sampled data) and we used a pdf window of size $W = 50$. The algorithm processed 457 data points per second on a 1.33 GHz PC in MATLAB/C under Linux, including the display of the ongoing segmentation, where one can observe the re-adaptation of past segmentation bounds and labels when new data becomes available.

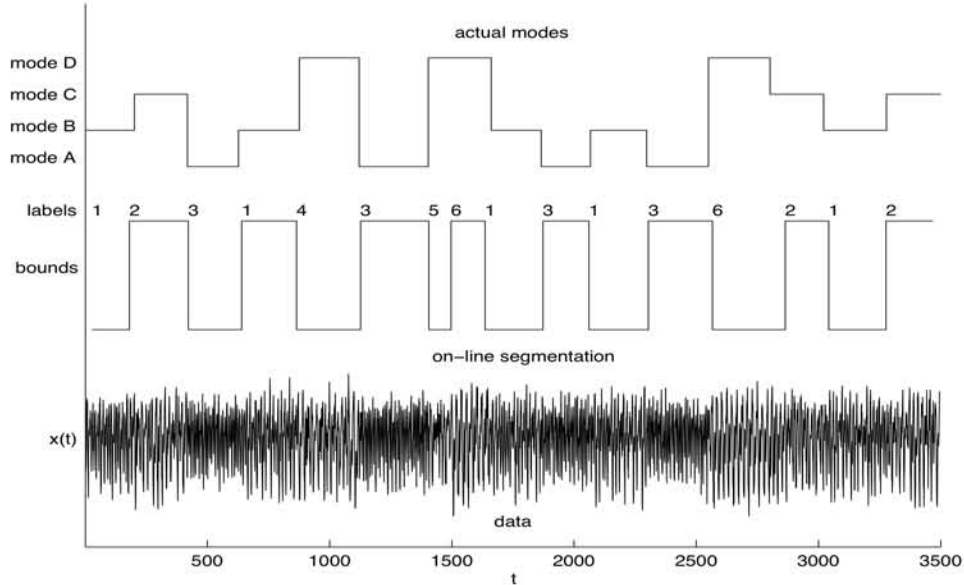

Figure 1: Segmentation of a switching Mackey-Glass time series with noise (bottom) that operates in four different modes (top). The on-line segmentation algorithm (middle), which receives the data points one by one, but not the mode information, yields correct segmentation bounds almost everywhere. The on-line labeler, however, assigns more labels (6) than desired (4), presumably due to the fact that the segments are very short and noisy.

The final segmentation is shown in Fig. 1. Surprisingly, the bounds of the segments are almost perfectly recovered from the very noisy data set. The only two exceptions are the third segment from the right, which is noticeably shorter than the original mode, and the segment in the middle, which is split in two by the algorithm. This split actually makes sense if one compares it with the data: there is a visible change in the signal characteristics at that point ($t \approx 1500$) even though the delay parameter was not modified there. This change is recorded by the algorithm since it operates in an unsupervised way.

The on-line labeling algorithm correctly assigns single labels to modes A, B, and C, but it assigns three labels (4, 5, and 6) to the segments of mode D, the most chaotic one. This is probably due to the small sample sizes (of the segments), in combination with the large amount of noise in the data.

## 4   Discussion

We presented an on-line method for the unsupervised segmentation and identification of sequential data, in particular from non-stationary switching dynamics. It is based on an HMM where the number of states varies dynamically as an effect of the way the incoming data is processed. In contrast to other approaches, it processes the data on-line and potentially even in real-time without training of any parameters. The method provides and updates a segmentation each time a new data point arrives. In effect, past segmentation bounds and labels are automatically re-adapted when new incoming data points are processed. The number of prototypes and labels that identify the segments is not fixed but determined by the

algorithm. We expect useful applications of this method in fields where complex non-stationary dynamics plays an important role, like, e.g., in physiology (EEG, MEG), climatology, in industrial applications, or in finance.

## Footnotes

[1]In our reported application we can process data at 1000 Hz (450 Hz including display) on a 1.33 GHz PC in MATLAB/C under Linux, which we expect is sufficient for a large number of applications.

[2]In that case, our further notation of time indices would refer to the subsampled data.

[3]We use $\vec{x}$ to denote a specific vector-valued *point* and $\mathbf{x}$ to denote a vector-valued *variable*.

[4]We developed an algorithm that computes an appropriate value for the hyperparameter $C$ from a sample set $\{\vec{x}_t\}$. Due to the limited space we will present that algorithm in a forthcoming publication [8].

## References

[1] Bellman, R. E. (1957). *Dynamic Programming*, Princeton University Press, Princeton, NJ.

[2] Bengio, Y., Frasconi, P. (1995). An Input Output HMM Architecture. In: *Advances in Neural Information Processing Systems* 7 (eds. Tesauro, Touretzky, Leen), Morgan Kaufmann, 427–434.

[3] Bengio, Y. (1999). Markovian Models for Sequential Data. *Neural Computing Surveys*, http://www.icsi.berkeley.edu/~jagota/NCS, 2:129–162.

[4] Bishop, C. M. (1995). *Neural Networks for Pattern Recognition*, Oxford Univ. Press, NY.

[5] Husmeier, D. (2000). Learning Non-Stationary Conditional Probability Distributions. *Neural Networks* 13, 287–290.

[6] Kehagias, A., Petridis, V. (1997). Time Series Segmentation using Predictive Modular Neural Networks. *Neural Computation* 9, 1691–1710.

[7] Kohlmorgen, J., Müller, K.-R., Rittweger, J., Pawelzik, K. (2000). Identification of Nonstationary Dynamics in Physiological Recordings, *Biol Cybern* 83(1), 73–84.

[8] Kohlmorgen, J., Lemm, S., *to appear*.

[9] Liehr, S., Pawelzik, K., Kohlmorgen, J., Müller, K.-R. (1999). Hidden Markov Mixtures of Experts with an Application to EEG Recordings from Sleep. *Theo Biosci* 118, 246–260.

[10] Mackey, M., Glass, L. (1977). Oscillation and Chaos in a Physiological Control System. *Science* 197, 287.

[11] Packard, N. H., Crutchfield J. P., Farmer, J. D., Shaw, R. S. (1980). Geometry from a Time Series. *Phys Rev Letters* 45, 712–716.

[12] Pawelzik, K., Kohlmorgen, J., Müller, K.-R. (1996). Annealed Competition of Experts for a Segmentation and Classification of Switching Dynamics. *Neural Computation* 8(2), 340–356.

[13] Rabiner, L. R. (1989). A Tutorial on Hidden Markov Models and Selected Applications in Speech Recognition, *Proceedings of the IEEE* 77(2), 257–286.

[14] Ramamurti, V., Ghosh, J. (1999). Structurally Adaptive Modular Networks for Non-Stationary Environments. *IEEE Tr. Neural Networks* 10(1), 152–160.
